# Linear Programming Analysis of Loopy Belief Propagation for Weighted Matching

**Sujay Sanghavi, Dmitry M. Malioutov and Alan S. Willsky**
Laboratory for Information and Decision Systems
Massachusetts Institute of Technology
Cambridge, MA 02139
{sanghavi,dmm,willsky}@mit.edu

## Abstract

Loopy belief propagation has been employed in a wide variety of applications with great empirical success, but it comes with few theoretical guarantees. In this paper we investigate the use of the max-product form of belief propagation for weighted matching problems on general graphs. We show that max-product converges to the correct answer if the linear programming (LP) relaxation of the weighted matching problem is tight and does not converge if the LP relaxation is loose. This provides an exact characterization of max-product performance and reveals connections to the widely used optimization technique of LP relaxation. In addition, we demonstrate that max-product is effective in solving practical weighted matching problems in a distributed fashion by applying it to the problem of self-organization in sensor networks.

## 1   Introduction

Loopy Belief Propagation (LBP) and its variants [6, 9, 13] have been shown empirically to be effective in solving many instances of hard problems in a wide range of fields. These algorithms were originally designed for exact inference (i.e. calculation of marginals/MAP estimates) in probability distributions whose associated graphical models are tree-structured. While some progress has been made in understanding their convergence and accuracy on general "loopy" graphs (see [8, 12, 13] and their references), it still remains an active research area.

In this paper we study the application of the widely used max-product form of LBP (or simply max-product (MP) algorithm), to the weighted matching problem. Given a graph $G = (V, E)$ with non-negative weights $w_e$ on its edges $e \in E$, the *weighted matching problem* is to find the heaviest set of mutually disjoint edges (i.e. a set of edges such that no two edges share a node). Weighted matching is a classic problem that has played a central role in computer science and combinatorial optimization, with applications in resource allocation, scheduling in communications networks [10], and machine learning [5]. It has often been perceived to be the "easiest non-trivial problem", and one whose analysis and solution has inspired methods (or provided insights) for a variety of other problems. Weighted matching thus naturally suggests itself as a good candidate for the study of convergence and correctness of algorithms like max-product.

Weighted matching can be naturally formulated as an integer program. The technique of *linear programming (LP) relaxation* involves replacing the integer constraints with linear inequality constraints. This relaxation is *tight* if the resulting linear program has an integral optimum. LP relaxation is *not* always tight for the weighted matching problem. The primary contribution of this paper is an exact characterization of max-product performance for the matching problem, which also establishes a link to LP relaxation. We show that (i) if the LP relaxation is tight then max-product

converges to the correct answer, and (ii) if the LP relaxation is not tight then max-product does not converge.

Weighted matching is a special case of the weighted $b$-matching problem, where there can be up to $b_i$ edges touching node $i$ (setting all $b_i = 1$ reduces to simple matching). All the results of this paper hold for the general case of $b$-matchings on arbitrary graphs. However, in the interests of clarity, we provide proofs only for the conceptually easier case of simple matchings where $b_i = 1$. The minor modifications needed for general $b$-matchings will appear in a longer publication. In prior work, Bayati et. al [2] established that max-product converges for weighted matching in *bipartite graphs*, and [5] extended this result to $b$-matching. These results are implied by our result[1], as for bipartite graphs, the LP relaxation is always tight.

In Section 2 we set up the weighted matching problem and its LP relaxation. We describe the max-product algorithm for weighted matching in Section 3. The main result of the paper is established in Section 4. Finally, in Section 5 we apply $b$-matching to a sensor-network self-organization problem and show that max-product provides an effective way to solve the problem in a distributed fashion.

## 2 Weighted Matching and its LP Relaxation

Suppose that we are given a graph $G$ with weights $w_e$, we also positive integers $b_i$ for each node $i \in V$. A $b$-*matching* is any set of edges such that the total number of edges in the set incident to any node $i$ is at most $b_i$. The *weighted $b$-matching problem* is to find the $b$-matching of largest weight. Weighted $b$-matching can be naturally formulated as the following integer program (setting all $b_i = 1$ gives an integer program for simple matching):

$$\textsf{IP}: \quad \max \quad \sum_{e \in E} w_e x_e,$$

$$\textsf{s.t.} \quad \sum_{e \in E_i} x_e \le b_i \quad \text{for all } i \in V,$$

$$x_e \in \{0, 1\} \quad \text{for all } e \in E$$

Here $E_i$ is the set of edges incident to node $i$. The *linear programming (LP) relaxation* of the above problem is to replace the constraint $x_e \in \{0, 1\}$ with the constraint $x_e \in [0, 1]$, for each $e \in E$. We denote the corresponding linear program by $\textsf{LP}$. Throughout this paper, we will assume that $\textsf{LP}$ has a *unique optimum*. The LP relaxation is said to be *tight* if the unique optimum is integral (i.e. one in which all $x_e \in \{0, 1\}$). Note that the LP relaxation is *not tight* in general. Apart from the bipartite case, the tightness of LP relaxation is a function of both the weights and the graph structure[2].

## 3 Max-Product for Weighted Matching

We now formulate weighted $b$-matching on $G$ as a MAP estimation problem by constructing a suitable probability distribution. This construction is naturally suggested by the form of the integer program $\textsf{IP}$. Associate a binary variable $x_e \in \{0, 1\}$ with each edge $e \in E$, and consider the following probability distribution:

$$p(x) \quad \propto \quad \prod_{i \in V} \psi(x_{E_i}) \prod_{e \in E} \exp(w_e x_e), \tag{1}$$

which contains a factor $\psi(x_{E_i})$ for each node $i \in V$, the value of which is $\psi(x_{E_i}) = 1$ if $\sum_{e \in E_i} x_e \le b_i$, and 0 otherwise. Note that we use $i$ to refer *both* to the nodes of $G$ and factors of $p$, and $e$ to refer both to the edges of $G$ and variables of $p$. The factor $\psi(x_{E_i})$ enforces the constraint that at most one edge incident to node $i$ can be assigned the value "1". It is easy to see that, for any $x$, $p(x) = \exp(\sum_e w_e x_e)$ if the set of edges $\{e | x_e = 1\}$ constitute a $b$-matching in $G$, and $p(x) = 0$ otherwise. Thus the max-weight $b$-matching of $G$ corresponds to the MAP estimate of $p$.

The factor-graph version of the max-product algorithm [6] passes messages between variables and the factors that contain them (for the formulation in (1), each variable is a member of exactly two factors). The output is an estimate $\hat{x}$ of the MAP of $p$. We now present the max-product update equations simplified for $p$ in (1). In the following $e$ and $(i, j)$ denote the same edge. Also, for two sets $A$ and $B$ the set difference is denoted by the notation $A - B$.

## Max-Product for Weighted Matching

**(INIT)** Set $t = 0$ and initialize each message to be uniform.

**(ITER)** Iteratively compute new messages until convergence as follows:

$$\text{Variable to Factor:} \qquad m_{e \to i}^{t+1}[x_e] = \exp(x_e w_e) \times m_{j \to e}^t[x_e]$$

$$\text{Factor to Variable:} \qquad m_{i \to e}^{t+1}[x_e] = \max_{x_{E_i - e}} \left\{ \psi(x_{E_i}) \prod_{e' \in E_i - e} m_{e' \to i}^t[x_{e'}] \right\}$$

Also, at each $t$ compute beliefs $n_e^t[x_e] = \exp(w_e x_e) \times m_{i \to e}^t[x_e] \times m_{j \to e}^t[x_e]$

**(ESTIM)** Upon convergence, output estimate $\hat{x}$: for each edge set $\hat{x}_e = 1$ if $n_e[1] > n_e[0]$, and $\hat{x}_e = 0$ otherwise.

**Remark:** If the degree $|E_i|$ of a node is large, the corresponding factor $\psi(x_{E_i})$ will depend on many variables. In general, for very large factors it is intractable to compute the "factor to variable" update (and even to store the factors in memory). However, for our problem the special form of $\psi$ makes this step easy even for large degrees: for each edge $e \in E_i$ compute $a_e = \max\left(1, \frac{m_{e \to i}^t[1]}{m_{e \to i}^t[0]}\right)$. Then, if all $b_i = 1$, we have that

$$m_{i \to e}^{t+1}[1] = \prod_{e' \in E_i - e} m_{e' \to i}^t[0] \qquad , \qquad m_{i \to e}^{t+1}[0] = \max_{e' \in E_i - e} a_{e'} \times \prod_{e' \in E_i - e} m_{e' \to i}^t[0]$$

The simplification for general $b$ is as follows: let $F_e \subset E_i - e$ be the set of $b_i$ variables in $E_i - e$ with the largest values of $a_e$, and let $G_e \subset E_i - e$ be the set of $b_i - 1$ variables with largest $a_e$. Then,

$$m_{i \to e}^{t+1}[1] = \prod_{e' \in G_e} a_{e'} \prod_{e' \in E_i - e} m_{e' \to i}^t[0] \qquad , \qquad m_{i \to e}^{t+1}[0] = \prod_{e' \in F_e} a_{e'} \prod_{e' \in E_i - e} m_{e' \to i}^t[0]$$

These updates are linear in the degree $|E_i|$.

### The Computation Tree for Weighted Matching

Our proofs rely on the computation tree interpretation [12, 11] of loopy max-product beliefs, which we now describe for the special case of simple matching ($b_i = 1$). Recall the variables of $p$ correspond to edges in $G$, and nodes in $G$ correspond to factors. For any edge $e$, the computation tree $T_e(1)$ at time 1 is just the edge $e$, the *root* of the tree. Both endpoints of the root are leaves. The tree $T_e(t)$ at time $t$ is generated from $T_e(t-1)$ by adding to each leaf of $T_e(t-1)$ a copy of each of its neighbors in $G$, except for the neighbor that is already present in $T_e(t-1)$. The weights of the edges in $T_e$ are copied from the corresponding edges in $G$.

Suppose $M$ is a matching on the original graph $G$, and $T_e$ is a computation tree. Then, the *image* of $M$ in $T_e$ is the set of edges in $T_e$ whose corresponding copy in $G$ is a member of $M$. We now illustrate the ideas of this section with a simple example.

**Example 1:** Consider the figure above. $G$ appears on the left, the numbers are the edge weights and the letters are node labels. The max-weight matching $M^* = \{(a, b), (c, d)\}$ is depicted in bold. In the center plot we show $T_{(a,b)}(4)$, the computation tree at time $t = 4$ rooted at edge $(a, b)$. Each node is labeled in accordance to its copy in $G$. The bold edges in the middle tree depict the matching which is the image of $M^*$ onto $T_{(a,b)}(4)$. The weight of this matching is 6.6, and it is easy to see that *any* matching on $T_{(a,b)}(4)$ that includes the root edge will have weight at most 6.6. On the right we depict $M$, the max-weight matching on the tree $T_{(a,b)}(4)$. $M$ has weight 7.3. In this example we see that even though $(a, b)$ is in the unique optimal matching in $G$, the beliefs at the root are such that $n_{(a,b)}^4[0] > n_{(a,b)}^4[1]$. Note also that the dotted edges are *not* an image of any matching in the original graph $G$. This example thus illustrates how "spurious" matchings in the computation tree can lead to incorrect beliefs, and estimates.

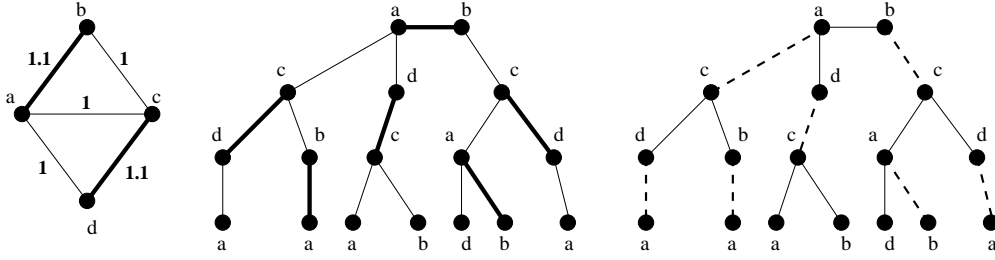

## 4 Main Result: Equivalence of LP Relaxation and Loopy Max-product

In this section we formally state the main result of this paper, and give an outline of the proofs.

**Theorem 1** *Let $G = (V, E)$ be a graph with nonnegative real weights $w_e$ on the edges $e \in E$. Assume the linear programming relaxation* LP *has a unique optimal solution. Then, the following holds:*

1. *If the LP relaxation is tight, i.e. if the unique solution is integral, then the max-product converges and the resulting estimate is the optimal matching.*

2. *If the LP relaxation is not tight, i.e. if the unique solution contains fractional values, then the max-product does not converge.*

The above theorem implies that LP relaxation and Max-product will both succeed, or both fail, on the same problem instances, and thus are equally powerful for the weighted matching problem. We now prove the two parts of the theorem. In the interest of brevity and clarity, the theorem and the proofs are presented for the conceptually easier case of simple matchings, in which all $b_i = 1$. Also, for the purposes of the proofs we will assume that "convergence" means that there exists a $\tau < \infty$ such that the maximizing assignment $\arg\max_{x_e} n_e^t(x_e)$ remains constant for all $t > \tau$.

**Proof of Part 1: Max-Product is as Powerful as LP Relaxation**

Suppose LP has an integral optimum. Consider now the linear-programming dual of LP, denoted below as DUAL.

$$\text{DUAL}: \quad \min \sum_{i \in V} z_i$$
$$\text{s.t.} \quad w_{ij} \leq z_i + z_j \quad \text{for all } (i,j) \in E,$$
$$z_i \geq 0 \quad \text{for all } i \in V$$

The following lemma states that the standard linear programming properties of complimentary slackness hold in the strict sense for the weighted matching problem (this is a special case of [3, ex. 4.20]).

**Lemma 1 (strict complimentary slackness)** *If the solution to* LP *is unique and integral, and $M^*$ is the optimal matching, then there exists an optimal dual solution $z$ to* DUAL *such that*

1. *For all $(i,j) \in M^*$, we have $w_{ij} = z_i + z_j$*

2. *There exists $\epsilon > 0$ such that for all $(i,j) \notin M^*$ we have $w_{ij} \leq z_i + z_j - \epsilon$*

3. *if no edge in $M^*$ is incident on node $i$, then $z_i = 0$*

4. *$z_i \leq \max_e w_e$ for all $i$*

Let $t \geq \frac{2w_{max}}{\epsilon}$, where $w_{max} = \max_e w_e$ is the weight of the heaviest edge, and $\epsilon$ is as in part 2 of Lemma 1 above. Suppose now that there exists an edge $e \notin M^*$ for which the belief at time $t$ is incorrect, i.e $n_e^t[1] > n_e^t[0]$. We now show that this leads to a contradiction.

Recall that $n_e^t[1] > n_e^t[0]$ means that there is a matching $M$ in $T_e(t)$ such that *(a)* the root $e \in M$, and *(b)* $M$ is a max-weight matching on $T_e(t)$. Let $M_T^*$ be the image of $M^*$ onto $T_e(t)$. By definition, $e \notin M_T^*$. From $e$, build an *alternating path* $P$ by successively adding edges as follows: first add $e$, then add all edges adjacent to $e$ that are in $M_T^*$, then all *their* adjacent edges that are in $M$, and so forth until no more edges can be added – this will occur either because no edges are available that maintain the alternating structure, or a leaf of $T_e(t)$ has been reached. Note that this will be a path, because $M$ and $M_T^*$ are matchings and so any node in $T_e(t)$ can have at most one edge adjacent to it in each of the two matchings.

For illustration, consider Example 1 of section 3. $M_T^*$ is in the center plot and $M$ is on the right. The above procedure for building $P$ would yield the path $adcabcda$ that goes from the left-most leaf to the right-most leaf. It is easy to see that this path alternates between edges in $M$ and $M_T^*$.

We now show that $w(P \cap M_T^*) > w(P \cap M)$. Let $z$ be the dual optimum corresponding to the $\epsilon$ above. Suppose first that neither endpoint of $P$ is a leaf of $T_e(t)$. Then, from parts 1 and 3 of Lemma 1 it follows that

$$w(P \cap M_T^*) \; = \; \sum_{(i,j) \in P \cap M_T^*} w_{ij} \; = \; \sum_{(i,j) \in P \cap M_T^*} z_i + z_j \; = \; \sum_{i \in P} z_i.$$

On the other hand, from part 2 of Lemma 1 it follows that

$$w(P \cap M) \; = \; \sum_{(i,j) \in P \cap M} w_{ij} \; \leq \; \sum_{(i,j) \in P \cap M} z_i + z_j - \epsilon \; = \; \left( \sum_{i \in P} z_i \right) - \epsilon |P \cap M|.$$

Now by construction the root $e \in P \cap S$, and hence $w(P \cap M_T^*) > w(P \cap M)$. A similar argument, with minor modifications, holds for the case when one or both endpoints of $P$ are leaves of $T_e$. For these cases we would need to make explicit use of the fact that $t \geq \frac{2w_{max}}{\epsilon}$.

We now show that $M$ cannot be an optimal matching in $T_e(t)$. We do so by "flipping" the edges in $P$ to obtain a matching with higher weight. Specifically, let $M' = M - (P \cap M) + (P \cap M_T^*)$ be the matching containing all edges in $M$ except the ones in $P$, which are replaced by the edges in $P \cap M_T^*$. It is easy to see that $M'$ is a matching in $T_e(t)$, and that $w(M') > w(M)$. This contradicts the choice of $M$, and shows that for $e \notin M^*$ the beliefs satisfy $n_e^t[1] \leq n_e^t[0]$ for all $t$ large enough. This means that the estimate has converged and is correct for $e$. A similar argument can be used to show that the max-product estimate converges to the correct answer for $e \in M^*$ as well. Hence max-product converges globally to the correct $M^*$.

**Proof of Part 2: LP Relaxation is as Powerful as Max-Product**

Suppose the optimum solution of LP contains fractional values. We now show that in this case max-product does not converge. As a first step we have the following lemma.

**Lemma 2** *If Max-Product converges, the resulting estimate is $M^*$.*

The proof of this lemma uses the result in [12], that states that if max-product converges then the resulting estimates are "locally optimal": the posterior probability of the max-product assignment can not be increased by changing values in any induced subgraph in which each connected component contains at most one loop. For the weighted matching problem this local optimality implies global optimality, because the symmetric difference of any two matchings is a union of disjoint paths and cycles. The above lemma implies that, for the proof of part 2 of the theorem, it is sufficient to show that max-product does not converge to the correct answer $M^*$. We do this by showing that for any given $\tau$, there exists a $t \geq \tau$ such that the max-product estimate at time $t$ will not be $M^*$.

We first provide a combinatorial characterization of when the LP relaxation is loose. Let $M^*$ be the max-weight matching on $G$. An alternating path in $G$ is a path in which every alternate edge is in $M^*$, and each node appears at most once. A *blossom* is an alternating path that wraps onto itself, such that the result is a single odd cycle $C$ and a path $R$ leading out of that cycle[3]. The importance of blossoms for matching problems is well-known [4]. A *bad blossom* is a blossom in which the edge weights satisfy

$$w(C \cap M^*) + 2w(R \cap M^*) \; < \; w(C - M^*) + 2w(R - M^*).$$

**Example:** On the right is a bad blossom: bold edges are in $M^*$, numbers are edge weights and alphabets are node labels. Cycle $C$ in this case is $abcdu$, and path $R$ is $cfghi$.

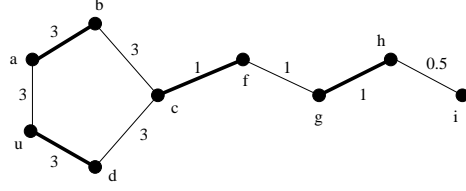

A *dumbbell* is an alternating path that wraps onto itself twice, such that the result is two disjoint odd cycles $C_1$ and $C_2$ and an alternating path $R$ connecting the two cycles. In a *bad dumbbell* the edge weights satisfy

$$w(C_1 \cap M^*) + w(C_2 \cap M^*) + 2w(R \cap M^*) \; < \; w(C_1 - M^*) + w(C_2 - M^*) + 2w(R - M^*).$$

**Example:** On the right is a bad dumbbell. Cycles $C_1$ and $C_2$ are $abcdu$ and $fghij$, and $(c, f)$ is the path $R$.

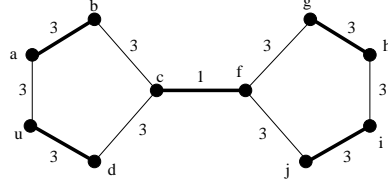

**Proposition 1** *If LP relaxation is loose, then in $G$ there exists either a bad blossom, or a bad dumbbell.*

*Proof.* The proof of this proposition will appear in a longer version of this paper. (It is also in the appendix submitted along with the paper).

Suppose now that max-product converges to $M^*$ by iteration $\tau$, and suppose also there exists a bad blossom $B_1$ in $G$. For an edge $e \in B_1 \cap M^*$ consider the computation tree $T_e(\tau + |V|)$ for $e$ at time $\tau + |V|$. Let $M$ be the optimal matching on the tree. From the definition of convergence, it follows that near the root $e$, $M$ will be the image of $M^*$ onto $T_e$: for any edge $e'$ in the tree at distance less than $|V|$ from the root, $e' \in M$ if and only if its copy in $G$ is in $M^*$.

This means that the copies in $T_e$ of the edges in $B_1$ will contain an alternating path $P$ in $T_e$: every alternate edge of $P$ will be in $M$. For the bad blossom example above, the alternating path is $ihgfcbaudcfghi$ (it will go once around the cycle and twice around the path of the blossom). Make a new matching $M'$ on $T_e(\tau + |V|)$ by "switching" the edges in this path: $M' = M - (M \cap P) + (P - M)$. Then, it is easy to see that

$$w(M) - w(M') = w(C \cap M^*) + 2w(R \cap M^*) - w(C - M^*) - 2w(R - M^*).$$

By assumption $B_1$ is a bad blossom, and hence we have that $w(M) < w(M')$, which violates the optimality of $M$. Thus, max-product does not converge to $M*$ if there exists a bad blossom. A similar proof precludes convergence to $M^*$ for the case when there is a bad dumbbell. It follows from Proposition 1 that if LP relaxation is loose, then max-product cannot converge to $M^*$.

## 5   Sensor network self-organization via $b$-matching

We now consider the problem of sensor network self-organization. Suppose a large number of low-cost sensors are deployed randomly over an area, and as a first step of any communication or remote sensing application the sensors have to organize themselves into a network [1]. The network should be connected, and robust to occasional failing links, but at the same time it should be sparse (i.e. have nodes with small degrees) due to severe limitations on power available for communication.

Simply connecting every pair of sensors that lie within some distance $d$ of each other (close enough to communicate reliably) may lead to large clusters of very densely connected components, and nodes with high degrees. Hence, sparser networks with fewer edges are needed [7]. The throughput of a link drops fast with distance, so the sparse network should mostly contain short edges. The sparsest connected network is achieved by a spanning tree solution. However, a spanning tree may have nodes with large degrees, and a single failed link disconnects it. An interesting set of sparse subgraph constructions with various tradeoffs addressing power efficiency in wireless networks is proposed in [7].

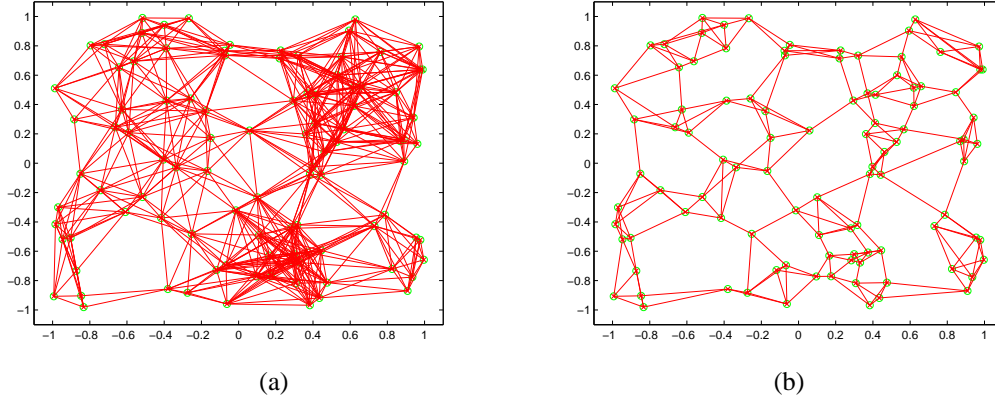

(a)                                                                                    (b)

Figure 1: Network with $N = 100$ nodes. (a) Nodes within $d = 0.5$ are connected by an edge. (b) Sparse network obtained by $b$-matching with $b = 5$.

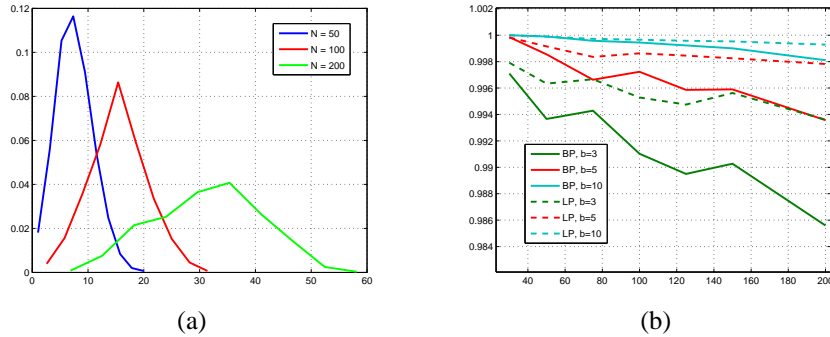

(a)                                                                                    (b)

Figure 2: (a) Histogram of node degrees versus node density. (b) Average fraction of the LP upper bound on optimal cost obtained using LP relaxation and max-product.

We consider using $b$-matching to find a sparse power-efficient subgraph. We assign edge weights to be proportional to the throughput of the link. For typical sensor network applications the received power (which can be used as a measure of throughput) decays as $d^{-p}$ with distance, where $p \in [2, 4]$. We set $p = 3$ for concreteness, and let the edge weights be $w_e = d_e^{-p}$. The $b$-matching objective is now to maximize the total throughput (received power) among sparse subgraphs with degree at most $b$. We use the max-product algorithm to solve weighted $b$-matching in a distributed fashion.

For our experiments we randomly disperse $N$ nodes in a square region $[-1, 1] \times [-1, 1]$. First we create the adjacency graph for nodes that are close enough to communicate, we set the threshold to be $d = 0.5$. In Figure 2(a) we plot the histogram over a 100 trials of resulting node degrees. Clearly, as $N$ increases, nodes have increasingly higher degrees.

Next we apply max-product (MP) and LP relaxation[4] to solve the $b$-matching objective. As we have established earlier, the performance of LP relaxation, and hence, of MP for b-matching depends on the existence of 'bad blossoms', i.e. odd-cycles where the weights on the edges are quite similar. We show in simulations that bad blossoms appear rarely for the random graphs and weights in our construction, and LP-relaxation and MP produce nearly optimal $b$-matchings. For the cases where LP relaxation has fractional edges, and MP has oscillating (or non-converged) edges, we erase them from the final matching and ensure that LP and MP solutions are valid matchings. Also, instead of comparing LP and MP costs to the optimal $b$-matching cost, we compare them to the LP upper bound on the cost (the cost of the fractional LP solution). This avoids the need to find optimal $b$-matchings.

In figure 1 we plot the dense adjacency graph for $N = 100$ nodes, and the much sparser $b$-matching subgraph with $b = 5$ obtained by MP. Now, consider figure 2(b). We plot the percentage of the LP

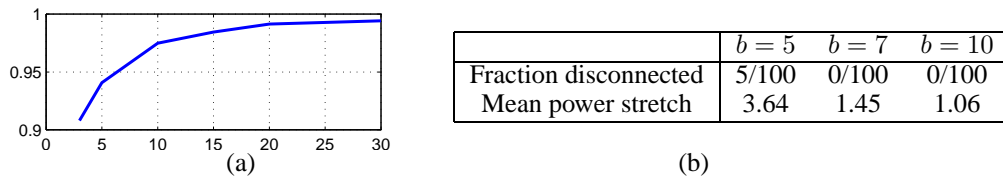

| | $b=5$ | $b=7$ | $b=10$ |
|---|---|---|---|
| Fraction disconnected | 5/100 | 0/100 | 0/100 |
| Mean power stretch | 3.64 | 1.45 | 1.06 |

(a)  (b)

Figure 3: (a) Average fraction of the LP upper bound on optimal cost obtained using $T$ iterations of max-product. (b) A table showing probability of disconnect, and the power stretch factor for $N = 100$ averaged over 100 trials.

upper bound obtained by MP and by rounded LP relaxation. It can be seen that both LP and MP produce nearly optimal $b$-matchings, with more than 98 percent of the optimal cost. The percentage decreases slowly with sensor density (with higher $N$), but improves for larger $b$. An important performance metric for sensor network self-organization is the power-stretch factor[5], which compares the weights of shortest paths in $G$ to weights of shortest paths in the sparse subgraph. In figure 3(b) we display the maximum power stretch factor over all pairs of nodes, averaged over 100 trials. For $b = 10$ there is almost no loss in power by using the sparse subgraph. A limitation of the $b$-matching solution is that connectedness of the subgraph is not guaranteed. In fact, for $b = 1$ it is always disconnected. However, as $b$ increases, the graph gets rarely disconnected. In figure 3(b) we display probability of disconnect over 100 trials. For $b = 10$ and $N = 100$ in a longer simulation, the sparse subgraph got disconnected twice over 500 trials.

In figure 3(a) we study the performance of MP versus the number of iterations. We run MP for a fixed number of iterations, remove oscillating edges to get a valid matching, and plot the average fraction of the LP upper bound that the solution gets. We set $b = 5$, and $N = 100$. Quite surprisingly, MP achieves a large percentage of the optimal cost even with as few as 3 iterations. After 20 this figure exceeds 99 percent.

## Footnotes

[1] However, [2] uses a graphical model which is different from ours to represent weighted matching.

[2] A simple example: $G$ is a cycle of length 3, all the $b_i = 1$. If all $w_e = 1$, LP relaxation is loose: setting each $x_e = \frac{1}{2}$ is the optimum. However, if instead the weights are $\{1, 1, 3\}$, then LP relaxation is tight.

[3]The path may be of zero length, in which case the blossom is just the odd cycle.

[4]LP is not practical for sensor networks, as it is not easily distributed.

[5]To compute the power-stretch the edges are weighted by $d^3$, i.e. the power needed to get a fixed throughput.

## References

[1] I.F. Akyildiz, W. Su, Y. Sankarasubramaniam, and E. Cayirci, "A survey on sensor networks," *IEEE Communications Magazine*, vol. 40, no. 8, pp. 102–114, Aug. 2002.

[2] M. Bayati, D. Shah, and M. Sharma, "Maximum weight matching via max-product belief propagation," in *ISIT*, Sept. 2005, pp. 1763 – 1767.

[3] D. Bertsimas and J. Tsitsiklis. *Linear Opitimization*. Athena Scientific, 1997.

[4] J. Edmonds, "Paths, trees and flowers," *Canadian Journal of Mathematics*, vol. 17, pp. 449–467, 1965.

[5] B. Huang and T. Jebara, "Loopy belief propagation for bipartite maximum weight b-matching," in *Artificial Intelligence and Statistics (AISTATS)*, March 2007.

[6] F. Kschischang, B. Frey, and H. Loeliger, "Factor graphs and the sum-product algorithm," *IEEE Transactions on Information Theory*, vol. 47, no. 2, pp. 498–519, Feb. 2001.

[7] X. Y. Li, P. J. Wan, Y. Wang, and O. Frieder, "Sparse power efficient topology for wireless networks," in *Proc. IEEE Hawaii Int. Conf. on System Sciences*, Jan. 2002.

[8] D. Malioutov, J. Johnson, and A. Willsky, "Walk-sums and belief propagation in Gaussian graphical models," *Journal of Machine Learning Research*, vol. 7, pp. 2031–2064, Oct. 2006.

[9] J. Pearl. *Probabilistic inference in intelligent systems*. Morgan Kaufmann, 1988.

[10] L. Tassiulas and A. Ephremides Stability properties of constrained queueing systems and scheduling policies for maximum throughput in multihop radio networks *IEEE Trans. on Automatic Control*, vol. 37, no. 12, Dec. 1992.

[11] S. Tatikonda and M. Jordan, "Loopy belief propagation and Gibbs measures," in *Uncertainty in Artificial Intelligence*, vol. 18, 2002, pp. 493–500.

[12] Y. Weiss and W. Freeman, "On the optimality of solutions of the max-product belief-propagation algorithm in arbitrary graphs," *IEEE Trans. on Information Theory*, vol. 47, no. 2, pp. 736–744, Feb. 2001.

[13] J. Yedidia, W. Freeman, and Y. Weiss. Understanding belief propagation and its generalizations. *Exploring AI in the new millennium*, pages 239–269, 2003.

